# Bayesian Network Score Approximation using a Metagraph Kernel

**Benjamin Yackley**
Department of Computer Science
University of New Mexico

**Eduardo Corona**
Courant Institute of Mathematical Sciences
New York University

**Terran Lane**
Department of Computer Science
University of New Mexico

## Abstract

Many interesting problems, including Bayesian network structure-search, can be cast in terms of finding the optimum value of a function over the space of graphs. However, this function is often expensive to compute exactly. We here present a method derived from the study of Reproducing Kernel Hilbert Spaces which takes advantage of the regular structure of the space of all graphs on a fixed number of nodes to obtain approximations to the desired function quickly and with reasonable accuracy. We then test this method on both a small testing set and a real-world Bayesian network; the results suggest that not only is this method reasonably accurate, but that the BDe score itself varies quadratically over the space of all graphs.

## 1 Introduction

The problem we address in this paper is, broadly speaking, function approximation. Specifically, the application we present here is that of estimating scores on the space of Bayesian networks as a first step toward a quick way to obtain a network which is optimal given a set of data. Usually, the search process requires a full recomputation of the posterior likelihood of the graph at every step, and is therefore slow. We present a new approach to the problem of approximating functions such as this one, where the mapping is of an object (the graph, in this particular case) to a real number (its BDe score). In other words, we have a function $f : \Gamma_n \to \mathbb{R}$ (where $\Gamma_n$ is the set of all directed graphs on $n$ nodes) from which we have a small number of samples, and we would like to interpolate the rest. The technique hinges on the set $\Gamma_n$ having a structure which can be factored into a Cartesian product, as well as on the function we approximate being smooth over this structure.

Although Bayesian networks are by definition acyclic, our approximation technique applies to the general directed-graph case. Because a given directed graph has $n^2$ possible edges, we can imagine the set of all graphs as itself being a Hamming cube of degree $n^2$ – a "metagraph" with $2^{n^2}$ nodes, since each edge can be independently present or absent. We say that two graphs are connected with an edge in our metagraph if they differ in one and only one edge. We can similarly identify each graph with a bit string by "unraveling" the adjacency matrix into a long string of zeros and ones. However, if we know beforehand an ordering on the nodes of our graph to which all directed graphs must stay consistent (to enforce acyclicness), then there are only $\binom{n}{2}$ possible edges, and the size of our metagraph drops to $2^{\binom{n}{2}}$. The same correspondence can then be made between these graphs and bit strings of length $\binom{n}{2}$.

Since the eigenvectors of the Laplacian of a graph form a basis for all smooth functions on the graph, then we can use our known sampled values (which correspond to a mapping from a subset of nodes on our metagraph to the real numbers) to interpolate the others. Despite the incredible size of the metagraph, we show that this problem is by no means intractable, and functions can in fact be approximated in polynomial time. We also demonstrate this technique both on a small network for which we can exhaustively compute the score of every possible directed acyclic graph, as well as on a larger real-world network. The results show that the method is accurate, and additionally suggest that the BDe scoring metric used is quadratic over the metagraph.

## 2 Spectral Properties of the Hypercube

### 2.1 The Kronecker Product and Kronecker Sum

The matrix operators known as the Kronecker product and Kronecker sum, denoted $\otimes$ and $\oplus$ respectively, play a key role in the derivation of the spectral properties of the hypercube. Given matrices $A \in \mathbb{R}^{i \times j}$ and $B \in \mathbb{R}^{k \times l}$, $A \otimes B$ is the matrix in $\mathbb{R}^{ik \times jl}$ such that:

$$A \otimes B = \begin{bmatrix} a_{11}B & a_{12}B & \cdots & a_{1j}B \\ a_{21}B & a_{22}B & & a_{2j}B \\ \vdots & & \ddots & \\ a_{j1}B & a_{j2}B & & a_{ij}B \end{bmatrix}$$

The Kronecker sum is defined over a pair of square matrices $A \in \mathbb{R}^{m \times m}$ and $B \in \mathbb{R}^{n \times n}$ as $A \oplus B = A \otimes I_n + I_m \otimes B$, where $I_n$ denotes an $n \times n$ identity matrix[8].

### 2.2 Cartesian Products of Graphs

The Cartesian product of two graphs $G_1$ and $G_2$, denoted $G_1 \times G_2$, is intuitively defined as the result of replacing every node in $G_1$ with a copy of $G_2$ and connecting corresponding edges together. More formally, if the product is the graph $G = G_1 \times G_2$, then the vertex set of $G$ is the Cartesian product of the vertex sets of $G_1$ and $G_2$. In other words, for any vertex $v_1$ in $G_1$ and any vertex $v_2$ in $G_2$, there exists a vertex $(v_1, v_2)$ in $G$. Additionally, the edge set of $G$ is such that, for any edge $(u_1, u_2) \to (v_1, v_2)$ in $G$, either $u_1 = v_1$ and $u_2 \to v_2$ is an edge in $G_2$, or $u_2 = v_2$ and $u_1 \to v_1$ is an edge in $G_1$.[7]

In particular, the set of hypercube graphs (or, identically, the set of Hamming cubes) can be derived using the Cartesian product operator. If we denote the graph of an $n$-dimensional hypercube as $Q_n$, then $Q_{n+1} = Q_n \times Q_1$, where the graph $Q_1$ is a two-node graph with a single bidirectional edge.

### 2.3 Spectral Properties of Cartesian Products

The Cartesian product has the property that, if we denote the adjacency matrix of a graph $G$ as $A(G)$, then $A(G_1 \times G_2) = A(G_1) \oplus A(G_2)$. Additionally, if $A(G_1)$ has $m$ eigenvectors $\phi_k$ and corresponding eigenvalues $\lambda_k$ (with $k = 1...m$) while $A(G_2)$ has $n$ eigenvectors $\psi_l$ with corresponding eigenvalues $\mu_l$ (with $l = 1...n$), then the full spectral decomposition of $A(G_1 \times G_2)$ is simple to obtain by the properties of the Kronecker sum; $A(G_1 \times G_2)$ will have $mn$ eigenvectors, each of them of the form $\phi_k \otimes \psi_l$ for every possible $\phi_k$ and $\psi_l$ in the original spectra, and each of them having the corresponding eigenvalue $\lambda_k + \mu_l$[2].

It should also be noted that, because hypercubes are all $k$-regular graphs (in particular, the hypercube $Q_n$ is $n$-regular), the form of the normalized Laplacian becomes simple. The usual formula for the normalized Laplacian is:

$$\tilde{L} = I - D^{-1/2}AD^{-1/2}$$

However, since the graph is regular, we have $D = kI$, and so

$$\tilde{L} = I - (kI)^{-1/2}A(kI)^{-1/2} = I - \frac{1}{k}A.$$

Also note that, because the formula for the combinatorial Laplacian is $L = D - A$, we also have $\tilde{L} = \frac{1}{k}L$.

The Laplacian also distributes over graph products, as shown in the following theorem.

**Theorem 1** *Given two simple, undirected graphs $G_1 = (V_1, E_1)$ and $G_2 = (V_2, E_2)$, with combinatorial Laplacians $L_{G_1}$ and $L_{G_2}$, the combinatorial Laplacian of the Cartesian product graph $G_1 \times G_2$ is then given by:*

$$L_{G_1 \times G_2} = L_{G_1} \oplus L_{G_2}$$

**Proof.**

$$L_{G_1} = D_{G_1} - A(G_1)$$
$$L_{G_2} = D_{G_2} - A(G_2)$$

Here, $D_G$ denotes the degree diagonal matrix of the graph $G$. Now, by the definition of the Laplacian,

$$L_{G_1 \times G_2} = D_{G_1 \times G_2} - A(G_1) \oplus A(G_2)$$

However, the degree of any vertex $uv$ in the Cartesian product is $\deg(u) + \deg(v)$, because all edges incident to a vertex will either be derived from one of the original graphs or the other, leading to corresponding nodes in the product graph. So, we have

$$D_{G_1 \times G_2} = D_{G_1} \oplus D_{G_2}$$

Substituting this in, we obtain

$$L_{G_1 \times G_2} = D_{G_1} \oplus D_{G_2} - A(G_1) \oplus A(G_2)$$
$$= D_{G_1} \otimes I_m + I_n \otimes D_{G_2} - A(G_1) \otimes I_m - I \otimes A(G_2)$$
$$= D_{G_1} \otimes I_m - A(G_1) \otimes I_m + I_n \otimes D_{G_2} - I_n \otimes A(G_2)$$

Because the Kronecker product is distributive over addition[8],

$$L_{G_1 \times G_2} = (D_{G_1} - A(G_1)) \otimes I_m + I_n \otimes (D_{G_2} - A(G_2))$$
$$= L_{G_1} \oplus L_{G_2}$$

Additionally, if $G_1 \times G_2$ is $k$-regular,

$$\tilde{L}_{G_1 \times G_2} = \tilde{L}_{G_1} \oplus \tilde{L}_{G_2} = \frac{1}{k}\left(L_{G_1} \oplus L_{G_2}\right)$$

∎

Therefore, since the combinatorial Laplacian operator distributes across a Kronecker sum, we can easily find the spectra of the Laplacian of an arbitrary hypercube through a recursive process if we just find the spectrum of the Laplacian of $Q_1$.

## 2.4   The Spectrum of the Hypercube $Q_n$

First, consider that

$$A(Q_1) = \begin{bmatrix} 0 & 1 \\ 1 & 0 \end{bmatrix}.$$

This is a $k$-regular graph with $k = 1$. So,

$$L_{Q_1} = I - \frac{1}{k}A(Q_1) = \begin{bmatrix} 1 & -1 \\ -1 & 1 \end{bmatrix}$$

Its eigenvectors and eigenvalues can be easily computed; it has the eigenvector $\begin{bmatrix} 1 \\ 1 \end{bmatrix}$ with eigenvalue 0 and the eigenvector $\begin{bmatrix} 1 \\ -1 \end{bmatrix}$ with eigenvalue 2. We can use these to compute the four eigenvectors of $L_{Q_2}$, the Laplacian of the 2-dimensional hypercube; $L_{Q_2} = L_{Q_1 \times Q_1} = L_{Q_1} \oplus L_{Q_1}$, so the four possible Kronecker products are $[1\,1\,1\,1]^T$, $[1\,1\,-1\,-1]^T$, $[1\,-1\,1\,-1]^T$, and $[1\,-1\,-1\,1]^T$, with corresponding eigenvalues 0, 1, 1, and 2 (renormalized by a factor of $\frac{1}{k} = \frac{1}{2}$ to take into account that our new hypercube is now degree 2 instead of degree 1; the combinatorial Laplacian would require no normalization). It should be noted here that an $n$-dimensional hypercube graph will have $2^n$ eigenvalues with only $n+1$ distinct values; they will be the values $\frac{2k}{n}$ for $k = 0...n$, each of which will have multiplicity $\binom{n}{k}$[4].

If we arrange these columns in the proper order as a matrix, a familiar shape emerges:

$$\begin{bmatrix} 1 & 1 & 1 & 1 \\ 1 & -1 & 1 & -1 \\ 1 & 1 & -1 & -1 \\ 1 & -1 & -1 & 1 \end{bmatrix}$$

This is, in fact, the Hadamard matrix of order 4, just as placing our original two eigenvectors side-by-side creates the order-2 Hadamard matrix. In fact, the eigenvectors of the Laplacian on a hypercube are simply the columns of a Hadamard matrix of the appropriate size; this can be seen by the recursive definition of the Hadamard matrix in terms of the Kronecker product:

$$H_{2^{n+1}} = H_{2^n} \otimes H_2$$

Recall that the eigenvectors of the Kronecker sum of two matrices are themselves all possible Kronecker products of eigenvectors of those matrices. Since hypercubes can be recursively constructed using Kronecker sums, the basis for smooth functions on hypercubes (i.e. the set of eigenvectors of their graph Laplacian) is the Hadamard basis. Consequently, there is no need to ever compute a full eigenvector explicitly; there is an explicit formula for a given entry of any Hadamard matrix:

$$(H_{2^n})_{ij} = (-1)^{\langle b_i, b_j \rangle}$$

The notation $b_x$ here means "the $n$-bit binary expansion of $x$ interpreted as a vector of 0s and 1s". This is the key to computing our kernel efficiently, not only because it takes very little time to compute arbitrary elements of eigenvectors, but because we are free to compute only the elements we need instead of entire eigenvectors at once.

## 3 The Metagraph Kernel

### 3.1 The Optimization Framework

Given the above, we now formulate the regression problem that will allow us to approximate our desired function at arbitrary points. Given a set of $k$ observations $\{y_i\}_{i=1}^k$ corresponding to nodes $x_i$ in the metagraph, we wish to find the $\hat{f}$ which minimizes the squared error between our estimate and all observed points and also which is a sufficiently smooth function on the graph to avoid overfitting. In other words,

$$\hat{f} = \arg\min_f \left\{ \frac{1}{k} \sum_{i=1}^k \|f(x_i) - y_i\|^2 + cf^T L^m f \right\}$$

The variable $m$ in this expression controls the type of smoothing; if $m = 1$, then we are penalizing first-differences (i.e. the gradient of the function). We will take $m = 2$ in our experiments, to penalize second-differences (the usual case when using spline interpolation)[6]. This problem can be formulated and solved within the Reproducing Kernel Hilbert Space framework[9]; consider the space of functions on our metagraph as the sum of two orthogonal spaces, one (called $\Omega_0$) consisting of functions which are not penalized by our regularization

term (which is $c\hat{f}L^m\hat{f}$), and one (called $\Omega_1$) consisting of functions orthogonal to those. In the case of our hypercube graph, $\Omega_0$ turns out to be particularly simple; it consists only of constant functions (i.e. vectors of the form $\mathbf{1}^T d$, where $\mathbf{1}$ is a vector of all ones). Meanwhile, the space $\Omega_1$ is formulated under the RKHS framework as a set of columns of the kernel matrix (denoted $K_1$). Consequently, we can write $\hat{f} = \mathbf{1}^T d + K_1 e$, and so our formulation becomes:

$$\hat{f} = \arg\min_f \left\{ \frac{1}{k} \sum_{i=1}^{k} \left\| (\mathbf{1}^T d + K_1 e)(x_i) - y_i \right\|^2 + c e^T K_1 e \right\}$$

The solution to this optimization problem is for our coefficients $d$ and $e$ to be linear estimates on $y$, our vector of observed values. In other words, there exist matrices $\Upsilon_d(c,m)$ and $\Upsilon_e(c,m)$, dependent on our smoothing coefficient $c$ and our exponent $m$, such that:

$$\hat{d} = \Upsilon_d(c,m)y$$
$$\hat{e} = \Upsilon_e(c,m)y$$
$$\hat{f} = \mathbf{1}^T \hat{d} + K_1 \hat{e} = \Upsilon(c,m)y$$

$\Upsilon(c,m) = \mathbf{1}^T \Upsilon_d(c,m) + K_1 \Upsilon_e(c,m)$ is the influence matrix[9] which provides the function estimate over the entire graph. Because $\Upsilon(c,m)$ is entirely dependent on the two matrices $\Upsilon_d$ and $\Upsilon_e$ as well as our kernel matrix, we can calculate an estimate for any set of nodes in the graph by explicitly calculating only those rows of $\Upsilon$ which correspond to those nodes and then simply multiplying that sub-matrix by the vector $y$. Therefore, if we have an efficient way to compute arbitrary entries of the kernel matrix $K_1$, we can estimate functions anywhere in the graph.

## 3.2 Calculating entries of $K_1$

First, we must choose an order $r \in \{1, 2...n\}$; this is equivalent to selecting the degree of a polynomial used to perform standard interpolation on the hypercube. The effect that $r$ will have on our problem will be to select the set of basis functions we consider; the eigenvectors corresponding to a given eigenvalue $\frac{2k}{n}$ are the $\binom{n}{k}$ eigenvectors which divide the space into identically-valued regions which are themselves $(n-k)$-dimensional hypercubes. For example, the 3 eigenfunctions on the 3-dimensional hypercube which correspond to the eigenvalue $\frac{2}{3}$ (so $k = 1$) are those which separate the space into a positive plane and a negative plane along each of the three axes. Because these eigenfunctions are all equivalent apart from rotation, there is no reason to choose one to be in our basis over another, and so we can say that the total number of eigenfunctions we use in our approximation is equal to $\sum_{k=0}^{r} \binom{n}{k}$ for our chosen value of $r$.

All eigenvectors can be identified with a number $l$ corresponding to its position in the natural-ordered Hadamard matrix; the columns where $l$ is an exact power of 2 are ones that alternate in identically-sized blocks of +1 and -1, while the others are element-wise products of the columns correponsing to the ones in $l$'s binary expansion. Therefore, if we use the notation $|x|_1$ to mean "the number of ones in the binary expansion of $x$", then choosing the order $r$ is equivalent to choosing a basis of eigenvectors $\phi_l$ such that $|l|_1$ is less than or equal to $r$. Therefore, we have:

$$(K_1)_{ij} = \sum_{1 \leq |l|_1 \leq r} \left( \frac{n}{2k} \right)^m H_{il} H_{jl}$$

Because $k$ is equal to $|l|_1$, and because we already have an explicit form for any $H_{xy}$, we can write

$$(K_1)_{ij} = \frac{1}{n} \sum_{1 \leq |l|_1 \leq r} \left( \frac{n}{2|l|_1} \right)^m (-1)^{<b_i,l>+<b_j,l>}$$

$$= \frac{1}{n} \sum_{k=1}^{r} \left( \frac{n}{2k} \right)^m \sum_{|l|_1=k} (-1)^{<b_i \dot{\vee} b_j, l>}$$

The $\dot\vee$ symbol here denotes exclusive-or, which is equivalent to addition mod 2 in this domain. The justification for this is that only the parity of the exponent (odd or even) matters, and locations in the bit strings $b_i$ and $b_j$ which are both zero or both one contribute no change to the overall parity. Notably, this shows that the value of the kernel between any two bit strings $b_i$ and $b_j$ is dependent only on $b_i \dot\vee b_j$, the key result which allows us to compute these values quickly. If we let $S_k(b_i, b_j) = \sum_{|l|_1=k}(-1)^{<b_i \dot\vee b_j, l>}$, there is a recursive formulation for the computation of $S_k(b_i, b_j)$ in terms of $S_{k-1}(b_i, b_j)$, which is the method used in the experiments due to its speed and feasability of computation.

# 4 Experiments

## 4.1 The 4-node Bayesian Network

The first set of experiments we performed were on a four-node Bayesian Network. We generated a random "base truth" network and sampled it 1000 times, creating a data set. We then created an exhaustive set of $2^6 = 64$ directed graphs; there are six possible edges in a four-node graph, assuming we already have some sort of node ordering that allows us to orient edges, and so this represented all possibilities. Because we chose the node ordering to be consistent with our base network, one of these graphs was in fact the correct network. We then gave each of the set of 64 graphs a log-marginal-likelihood score (i.e. the BDe score) based on the generated data. As expected, the correct network came out to have the greatest likelihood. Additionally, computation of the Rayleigh quotient shows that the function is a globally smooth one over the graph topology. We then performed a set of experiments using the metagraph kernel.

### 4.1.1 Randomly Drawn Observations

First, we partitioned the set of 64 observations randomly into two groups. The training group ranged in size from 3 to 63 samples, with the rest used as the testing group. We then used the training group as the set of observations, and queried the metagraph kernel to predict the values of the networks in the testing group. We repeated this process 50 times for each of the different sizes of the training group, and the results averaged to obtain Figure 1. Note that order 3 performs the best overall for large numbers of observations, overtaking the order-2 approximation at 41 values observed and staying the best until the end. However, order 1 performs the best for small numbers of observations (perhaps due to overfitting errors caused by the higher orders) and order 2 performs the best in the middle. The data suggests that the proper order to which to compute the kernel in order to obtain the best approximations is a function of both the size of the space and the number of observations made within that space.

### 4.1.2 Best/worst-case Observations

Secondly, we performed experiments where the observations were obtained from networks which were in the neighborhood around the known true maximum, as well as ones from networks which were as far from it as possible. These results are Figures 2 and 3. Despite small differences in shape, the results are largely identical, indicating that the distribution of the samples throughout $\Gamma_n$ matters very little.

## 4.2 The ALARM Network

The ALARM Bayesian network[1] contains 37 nodes, and has been used in much Bayes-net-related research[3]. We first generated data according to the true network, sampling it 1000 times, then generated random directed graphs over the 37 nodes to see if their scores could be predicted as well as in the smaller four-node case. We generated two sets of these graphs: a set of 100, and a set of 1000. We made no attempt to enforce an ordering; although the graphs were all acyclic, we placed no assumption on the graphs being consistent with the same node-ordering as the original. The scores of these sets, calculated using the data drawn from the true network, served as our observed data. We then used the kernel to

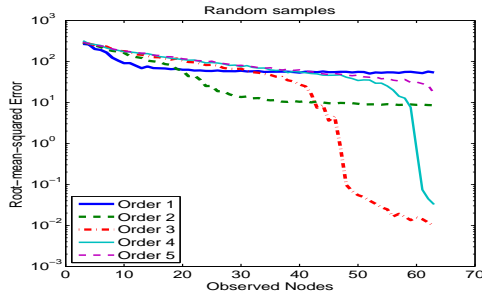
(a) Figure 1: Randomly-drawn Samples

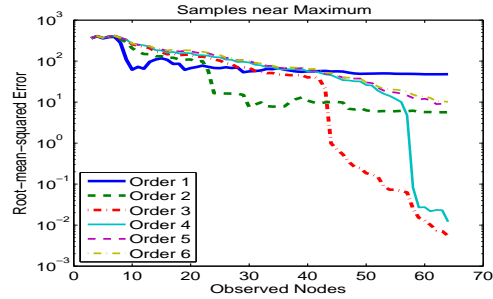
(b) Figure 2: Samples drawn near maximum

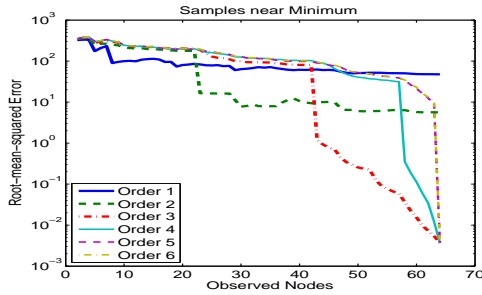
(c) Figure 3: Samples drawn near minimum

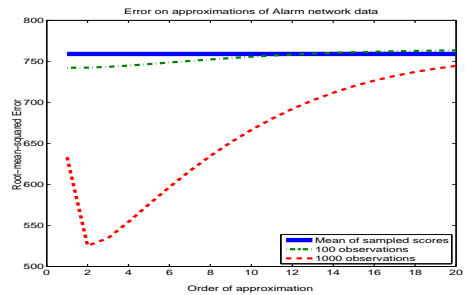
(d) Figure 4: Samples from ALARM network

approximate, given the observed scores, the score of an additional 100 randomly-generated graphs, with the order of the kernel varying from 1 to 20. The results, with root-mean-squared error plotted against the order of the kernel, are shown in Figure 4. Additionally, we calculated a baseline by taking the mean of the 1000 sampled scores and calling that the estimated score for every graph in our testing set.

The results show that the metagraph approximation method performs significantly better than the baseline for low orders of approximation with higher amounts of sampled data. This makes intuitive sense; the more data there is, the better the approximation should be. Additionally, the spike at order 2 suggests that the BDe score itself varies quadratically over the metagraph. To our knowledge, we are the first to make this observation. In current work, we are analyzing the BDe in an attempt to analytically validate this empirical observation. If true, this observation may lead to improved optimization techniques for finding the BDe-maximizing Bayesian network. Note, however, that, even if true, exact optimization is still unlikely to be polynomial-time because the quadratic form is almost certainly indefinite and, therefore, NP-hard to optimize.

# 5 Conclusion

Functions of graphs to real numbers, such as the posterior likelihood of a Bayesian network given a set of data, can be approximated to a high degree of accuracy by taking advantage of a hybercubic "metagraph" structure. Because the metagraph is regular, standard techniques of interpolation can be used in a straightforward way to obtain predictions for the values at unknown points.

# 6 Future Work

Although this technique allows for quick and accurate prediction of function values on the metagraph, it offers no hints (as of yet) as to where the maximum of the function might be. This could, for instance, allow one to generate a Bayesian network which is likely to be close to optimal, and if true optimality is required, that approximate graph could be used

as a starting point for a stepwise method such as MCMC. Even without a direct way to find such an optimum, though, it may be worth using this approximation technique inside an MCMC search instead of the usual exact-score computation in order to quickly converge on a something close to the desired optimum.

Also, many other problems have a similar flavor. In fact, this technique should be able to be used unchanged on any problem which involves the computation of a real-valued function over bit strings. For other objects, however, the structure is not necessarily a hypercube. For example, one may desire an approximation to a function of permutations of some number of elements to real numbers. The set of permutations of a given number of elements, denoted $S_n$, has a similarly regular structure (which can be seen as a graph in which two permutations are connected if a single swap leads from one to the other), but not a hypercubic one. The structure-search problem on Bayes Nets can also be cast as a search over orderings of nodes alone[5], so a way to approximate a function over permutations would be useful there as well.

Other domains have this ability to be turned into regular graphs – the integers mod $n$ with edges between numbers that differ by 1 form a loop, for example. It should be possible to apply a similar trick to obtain function approximations not only on these domains, but on arbitrary Cartesian products of them. So, for instance, remembering that the directions of the edges of Bayesian network are completely specified given an ordering on the nodes, the network structure search problem on $n$ nodes can be recast as a function approximation over the set $S_n \times Q_{\binom{n}{2}}$. Many problems can be cast into the metagraph framework; we have only just scratched the surface here.

## Acknowledgments

The authors would like to thank Curtis Storlie and Joshua Neil from the UNM Department of Mathematics and Statistics, as well as everyone in the Machine Learning Reading Group at UNM. This work was supported by NSF grant #IIS-0705681 under the Robust Intelligence program.

## References

[1] I. Beinlich, H.J. Suermondt, R. Chavez, G. Cooper, et al. The ALARM monitoring system: A case study with two probabilistic inference techniques for belief networks. *Proceedings of the Second European Conference on Artificial Intelligence in Medicine*, 256, 1989.

[2] D.S. Bernstein. *Matrix Mathematics: Theory, Facts, and Formulas with Application to Linear Systems Theory*. Princeton University Press, 2005.

[3] D.M. Chickering, D. Heckerman, and C. Meek. A Bayesian approach to learning Bayesian networks with local structure. *UAI'97*, pages 80–89, 1997.

[4] Fan R. K. Chung. *Spectral Graph Theory*. Conference Board of the Mathematical Sciences. AMS, 1997.

[5] N. Friedman and D. Koller. Being Bayesian about network structure. *Machine Learning*, 50(1-2):95–125, 2003.

[6] Chong Gu. *Smoothing Splines ANOVA Models*. Springer Verlag, 2002.

[7] G. Sabidussi. Graph multiplication. *Mathematische Zeitschrift*, 72(1):446–457, 1959.

[8] Kathrin Schacke. On the kronecker product. Master's thesis, University of Waterloo, 2004.

[9] Grace Wahba. *Spline Models for Observational Data*. CBMS-NSF Regional Conference Series in Applied Mathematics. SCIAM, 1990.
